# Exploiting Syllable Structure
# in a Connectionist Phonology Model

David S. Touretzky   Deirdre W. Wheeler
School of Computer Science
Carnegie Mellon University
Pittsburgh, PA 15213-3890

## Abstract

In a previous paper (Touretzky & Wheeler, 1990a) we showed how adding a clustering operation to a connectionist phonology model produced a parallel processing account of certain "iterative" phenomena. In this paper we show how the addition of a second structuring primitive, *syllabification*, greatly increases the power of the model. We present examples from a non-Indo-European language that appear to require rule ordering to at least a depth of four. By adding syllabification circuitry to structure the model's perception of the input string, we are able to handle these examples with only two derivational steps. We conclude that in phonology, derivation can be largely replaced by structuring.

## 1 Introduction

In linguistics a *grammar* is an abstract formal system describing a language. The term *psycho-grammar* has been suggested for systems that express the linguistic knowledge that actually exists in speakers' heads (George, 1989). Psycho-grammars may differ from grammars as a result of performance demands, limited memory capacity, or other aspects of mental representations. Psycho-grammars are still somewhat abstract, in that they are concerned with mental rather than physical phenomena. The term *physio-grammar* (George, 1989) refers to the the physical representation of grammatical knowledge in neural structures, such as (perhaps) synapse strengths. Detailed proposals for physio-grammars do not yet exist; the field of neurolinguistics is insufficiently advanced to support such proposals at present.

We are developing a theory of phonology that is compatible with gross constraints on neural processing and cognitive plausibility. Our research, then, is on the construction of psycho-grammars at the phonological level. We use a connectionist model to demonstrate

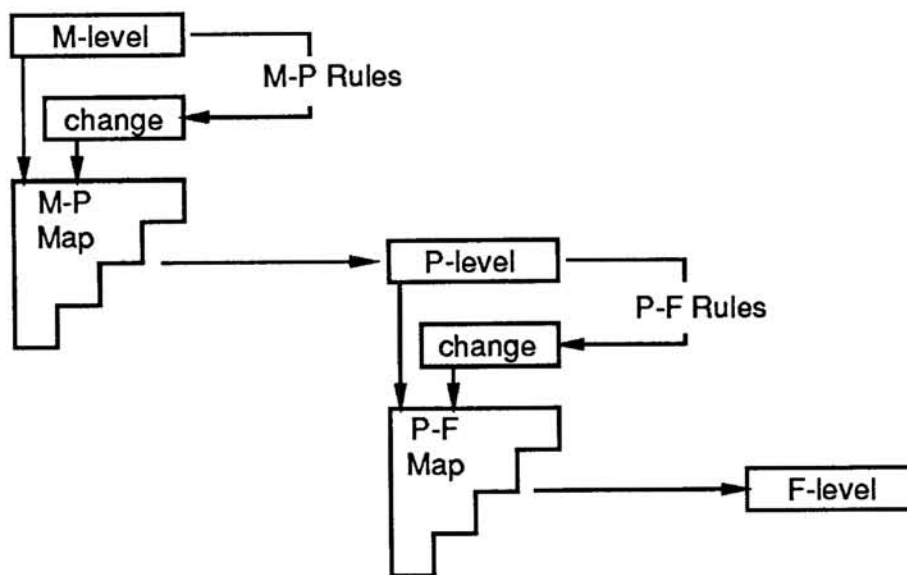

Figure 1: Structure of the model.

the computational feasibility of the psycho-grammar architecture we propose. In this paper we show how the addition of syllabification as a primitive operation greatly increases the scope and power of the model at little computational cost.

## 2  Structure of the Model

Our model, shown in Figure 1, has three levels of representation. Following Lakoff (1989), they are labeled M, P, and F. The M, or morpho-phonemic level, is a sequence of phonemes constructed by concatenating abstract underlying representations of morphemes. The P, or phonemic level, is an intermediate representation that is constrained to hold syllabically well-formed strings. The F, or phonetic level, is the surface level representation: a sequence of phonetic segments. Derivations are performed by mapping strings from M to P level, and then from P to F level, under the control of a set of language-specific rules. These rules alter the mapping in various ways to effect processes such as voicing assimilation and vowel harmony.

The model has a number of important constraints. Rules at a given level (M-P or P-F) apply in a single parallel step during the mapping from one level to the next. There is no iterative rule application. "Iterative" processes are instead handled by a parallel clustering mechanism described in Touretzky & Wheeler (1990a,1991). The connectionist implementation uses limited-depth, strictly feed-forward circuitry, so the model has minimal computational complexity.

Another very important constraint is that only two levels of derivation are provided, M-P and P-F, so there is no room for the long chains of ordered rules that other phonological theories permit. However, in standard analyses some languages appear to require long rule chains. The problem for those who want to eliminate such chains on grounds of cognitive implausibility[1] is to reformulate existing linguistic analyses to account for the data in some

| / hro+aht ũ/ | | "he has disappeared" |
|---|---|---|
| hro | ht ũ | *vowel deletion* |
| hró | ht ũ | *stress* |
| ró | ht ũ | *initial h-deletion* |
| ró | hdũ | *pre-son. voicing* |
| [ ró | hdũ] | |

| / ʌ̃+k+hrek+?/ | | "I will push it" |
|---|---|---|
| ʌ̃́ | k hrek ? | *stress* |
| ʌ̃́ | k hreke? | *epenthesis* |
| ʌ̃́ | k hrege? | *pre-son. voicing* |
| [ʌ̃́ | k hrege?] | |

Figure 2: Two Mohawk derivations.

other way. This is not always easy to do, especially in our model, which is more tightly constrained than either the Goldsmith or Lakoff proposals. Such reformulations help us to see how psycho-grammar diverges from grammar when computational constraints are taken into consideration.

## 3   A Problem From Mohawk

In Mohawk, an American Indian language, stress is placed on the penultimate syllable of a word. Since there are processes in Mohawk that add and delete vowels from words, their interaction with the stress rule is problematic. Figure 2 shows two Mohawk derivations in a standard generative account.[2] The first example shows us that vowel deletion must precede stress assignment. The penultimate vowel /a/ in the underlying form does not appear in the surface form of the word. Instead stress is assigned to the preceding vowel, /o/, which is is the penultimate vowel in the surface form. The second example shows that stress assignment must precede vowel epenthesis (insertion), because the epenthetic /e/ that appears in the surface form is not counted when determining the penultimate vowel. Since the epenthetic /e/ is also the trigger for presonorant voicing in this example, we see that voicing must be ordered after vowel epenthesis. Together these two examples indicate the following rule ordering: Vowel deletion < Stress < Epenthesis < Pre-sonorant voicing. But this is a depth of four, and our model permits only two levels of derivation. We therefore must produce an alternative account of these four processes that requires fewer derivations. To do so, we rely on three features of the model: parallel rule application, multi-level representations, and a structuring primitive: syllabification.

## 4   Representation of Syllable Structure

Most insertion and deletion operations are syllabically-motivated (Itô, 1989). By adding a syllabification mechanism to our model, we can replace certain derivational (string-rewriting) steps with more constrained and perhaps cognitively less taxing structuring steps. Linguists represent syllables as tree structures, as in the left portion of Figure 3. The nucleus of the syllable is normally a vowel. Any preceding consonants form the onset, and any following consonants the coda. The combined nucleus and coda make up the rime. In the middle portion of Figure 3 the syllabic structure of the English word "tokens" (phonetic transcription [tokɛnz]) is shown in this hierarchically structured form. The right portion shows how we encode the same information in our model using a set of onset, nucleus,

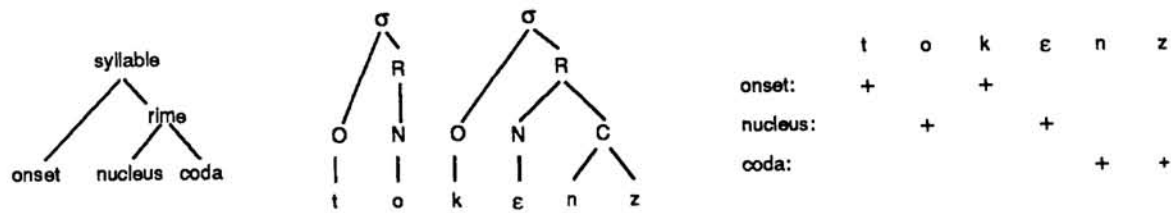

Figure 3: Representations for syllable structure.

| M: | hr oaht ũ | |
|---|---|---|
| onset: | ++ + | |
| nucleus: | + + | *vowel del.* |
| coda: | + | |
| | | *stress (M-P)* |
| P: | hr ó ht ũ | |
| | | *h-del.; pre-son.* |
| F: | r ó hdũ | *voicing (P-F)* |

| M: | ʌ̃khr ek ? | |
|---|---|---|
| onset: | ++ + | *epenthesis* |
| nucleus: | + + | |
| coda: | + + | |
| | | *stress (M-P)* |
| P: | ʌ̃khr eke? | |
| | | *pre-son.* |
| F: | ʌ̃khr ege? | *voicing (P-F)* |

Figure 4: Our solution to the Mohawk problem.

and coda bits, or ONC bits for short. We have no explicit representation for rimes, but this could be added if necessary.

In Mohawk, the vowel deletion and epenthesis processes are both syllabically motivated. Vowel deletion enforces a constraint against branching nuclei.[3] Epenthesis inserts a vowel to break up a word-final consonant cluster (/k/ followed by glottal stop /?/) that would be an illegal syllable coda. Our contention is that syllabification operates on the M-level string by setting the associated ONC bits in such a way that the P-level string will be syllabically well-formed. The ONC bits share control with the M-P rules of the mapping from M to P level.

Every M-level segment must have one of its ONC bits set in order to be mapped to P-level. Thus, the syllabifier can cause a vowel to be deleted at P simply by failing to set its nucleus bit, as occurs for the /a/ in /hroahtũ/ in Figure 4. For the /ʌ̃khrek?/ example, note in Figure 4 that the /k/ has been marked as an onset by the syllabifier and the /?/ as a coda; there is no intervening nucleus. This automatically triggers an insertion by the M-P map, so that a vowel will appear between these two segments at P-level. The vowel chosen is the default or "unmarked" vowel for that particular language; for Mohawk it is /e/. For further details of the syllabification algorithm, see Touretzky & Wheeler (1990b).

The left half of Figure 5 shows our formulation of the Mohawk stress rule, which assigns stress to the penultimate nucleus of a word. Rather than looking directly at the M-level buffer, the rule looks at the "projection" of the nucleus tier. By this we mean the M-level substring consisting of those segments whose nucleus bit is set. The # symbol indicates a word boundary. Since vowels deleted by the syllabifier have no nucleus bit set, and

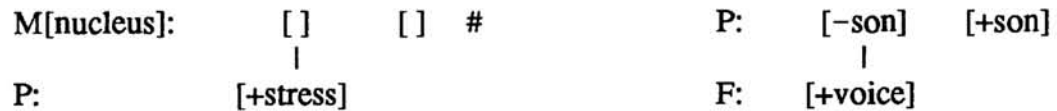

Figure 5: Rules for Mohawk stress (M-P) and presonorant voicing (P-F).

epenthetic vowels that will be inserted by the syllabifier have no nucleus bit at M-level, insertion and deletion processes can proceed in parallel with stress assignment. At P-level, all that's left to be done in this example is pre-sonorant voicing, handled by the P-F rule shown in the right half of the figure.

# 5   More Complex Stress Rules

In Mohawk, stress falls on the penultimate syllable regardless of the internal structure of the syllable. This stress assignment rule is quite simple compared to some other languages. For example, "quantity sensitive" languages make distinctions among syllable types for purposes of stress assignment. A syllable consisting of an optional onset and a single, short vowel in the rime is normally said to be "light," while syllables with codas and/or long vowels (often represented as double nuclei) are designated "heavy," and typically attract stress. Thus, for example, in Aguacatec Mayan (Hayes, 1981) stress falls on the rightmost syllable with a long vowel, otherwise the final syllable.

In order to account for syllable weight distinctions we introduce an additional level of representation, as illustrated in Figure 6 using C and V to represent consonants and vowels, respectively. The "mora" bit is activated for all segments that contribute to syllable weight in the language. In this particular language only vowels are important for determining the weight of syllables, so the mora bit is activated for all and only the vocalic segments. Once moras have been identified, universal principles come into play, and bits for "syllable" and "heavy syllable" are set. The syllable bit is activated for the first of a sequence of one or more moras; the heavy syllable bit is activated for syllables containing two or more moras. With this enriched representation, the stress patterns of quantity-sensitive languages can be straightforwardly generated. To stress the last heavy syllable, we assign [+stress] to segments on the heavy syllable tier that have word boundaries to their right. (Word boundaries must be projected down to the heavy syllable tier for this purpose.)

Languages like Yana (Hayes, 1981), in which both long vowels and codas make syllables heavy, have a slightly different representation at the mora level. In these languages, coda consonants as well as vocalic segments trigger the activation of the mora bit, as illustrated in Figure 7. Here again, while specification of what counts as a mora is a language-specific parameter, once the mora bits are set the syllable and heavy syllable representations follow from universal principles. The mora bit is activated for any segment which has either the nucleus or coda bit set, essentially collapsing the nucleus and coda tiers. The Yana stress rule targets the leftmost heavy syllable in a word, no matter how far it might occur from the initial word boundary, or the first syllable if none are heavy. The latter case requires a separate rule with a slightly more complex environment; rules of this form are discussed in Wheeler & Touretzky (1991).

| | # | C | V | C | V | C | V | V | C | V | C | V | V | C | # |
|---|---|---|---|---|---|---|---|---|---|---|---|---|---|---|---|
| onset | | + | | + | | + | | | + | | + | | | | |
| nucleus | | | + | | + | | + | + | | + | | + | + | | |
| coda | | | | | | | | | | | | | | + | |
| mora | | | + | | + | | + | + | | + | | + | + | | |
| syllable | | | + | | + | | + | | | + | | + | | | |
| heavy syllable | | | | | | | + | | | | | + | | | |

Figure 6: Long vowels make syllables heavy in Aguacatec Mayan.

| | # | C | V | C | V | C | C | V | C | V | V | C | V | V | C | # |
|---|---|---|---|---|---|---|---|---|---|---|---|---|---|---|---|---|
| onset | | + | | + | | | + | | + | | | + | | | | |
| nucleus | | | + | | + | | | + | | + | + | | + | + | | |
| coda | | | | | | + | | | | | | | | | + | |
| mora | | | + | | + | + | | + | | + | + | | + | + | + | |
| syllable | | | + | | + | | | + | | + | | | + | | | |
| heavy syllable | | | | | + | | | | | + | | | + | | | |

Figure 7: Long vowels or codas make syllables heavy in Yana.

# 6   Discussion

For the linguist, it is interesting to see how structuring operations such as clustering and syllabification can take some of the pressure off derivation, thereby allowing strict limits to be maintained on derivational depth. But what is the significance of this work for connectionists? Unlike most other attempts to model phonological processes in neural networks, we demonstrate the influence computational modeling can have on the development of a linguistic theory. In designing a system for expressing linguistic processes, there must be some sort of cost metric to determine which operations are computationally feasible and which are not. A connectionist implementation provides a natural cost metric: size (depth, fanout, component count) of the required threshold logic circuity.

It is doubtful that the structure of our model corresponds to that of some cortical language area, and we reject any simplistic analogy between threshold logic units and neurons. Using circuit complexity as a cost metric can be independently justified on grounds of simplicity and theoretical elegance. If one measures cost in some more abstract way, there is a danger that computationally expensive mechanisms may lurk beneath the grammar's apparent simplicity. An example is the local rule ordering proposal of Anderson (1974), in which explicit rule ordering is eliminated by introducing a much more complex mechanism for determining, on a case-by-case basis, the order in which rules should apply.

If the mental representation of utterances is fundamentally different from the discrete symbolic form we've assumed,[4] we may be using the wrong cost metric for determining cognitive plausibility. However, we are constrained, like everyone else, to work within the computational frameworks that are presently available.

There remains the question of why structuring should be preferred over derivation. First, since some mutation processes are sensitive to syllabic structure, this information would have to be computed even if insertions and deletions weren't handled by the syllabifier. Second, structuring is a highly constrained operation; it merely annotates an existing string to reflect constituency relationships, whereas derivations can make arbitrary changes to a string. We therefore assume that derivations have a higher cognitive cost, despite the fact that they can be computed fairly efficiently in our model by the mapping matrix described in Touretzky & Wheeler (1991). Finally, adding extra derivational levels increases the difficulty of phonological rule induction, a topic of current research.

## Acknowledgements

This work was sponsored by a grant from the Hughes Aircraft Corporation, by National Science Foundation grant EET-8716324, and by the Office of Naval Research under contract number N00014-86-K-0678.

## Footnotes

[1]Here we are referring to Goldsmith (1990) and Lakoff (1989), as well as our own work.

[2]These examples, derived from Halle & Clements (1983), are cited in Lakoff (1989). We thank Marianne Mithun (p.c.) for correcting an error in the original data.

[3]This constraint is not shared by all languages. Furthermore, deletion is only one possible solution; another would be to insert a consonant or glide, such as /w/, to separate the vowels into different syllables. Each language makes its own choices about how constraint violations are to be repaired.

[4]For example: if phonetic strings turn out to be represented in the brain as chaotic trajectories in a high dimensional dynamical system, or something equally exotic.

## References

Anderson, S. R. (1974) *The Organization of Phonology.* New York: Academic Press.

George, A. (1989) How not to become confused about linguistics. In A. George (ed.), *Reflections on Chomsky*, 90-110. Oxford, UK: Basil Blackwell.

Goldsmith, J. A. (1990) *Autosegmental and Metrical Phonology.* Oxford, UK: Basil Blackwell.

Halle, M., and Clements, G. N. (1983) *Problem Book in Phonology: A Workbook for Introductory Courses in Linguistics and Modern Phonology.* Cambridge, MA: The MIT Press.

Hayes, B. (1981) *A Metrical Theory of Stress Rules.* Doctoral dissertation, MIT, Cambridge, MA.

Itô, J. (1989) A prosodic theory of epenthesis. *Natural Language and Linguistic Theory*, 7(2), 217-259.

Lakoff, G. (1989) Cognitive phonology. Draft of paper presented at the UC-Berkeley Workshop on Constraints vs. Rules, May 1989.

Touretzky, D. S., and Wheeler, D. W. (1990a) A computational basis for phonology. In D. S. Touretzky (ed.), *Advances in Neural Information Processing Systems 2*, 372-379. San Mateo, CA: Morgan Kaufmann.

Touretzky, D. S., and Wheeler, D. W. (1990b) Two derivations suffice: the role of syllabification in cognitive phonology. In C. Tenny (ed.), *The MIT Parsing Volume, 1989-1990*, 21-35. MIT Center for Cognitive Science, Parsing Project Working Papers 3.

Touretzky, D. S., and Wheeler, D. W. (1991) Sequence manipulation using parallel mapping networks. *Neural Computation* 3(1):98-109.

Wheeler, D. W., and Touretzky, D. S. (1991) From syllables to stress: a cognitively plausible model. In K. Deaton, M. Noske, and M. Ziolkowski (eds.), *CLS 26-II: Papers from the Parasession on The Syllable in Phonetics and Phonology, 1990.* Chicago Linguistic Society.